# Meta-Gaussian Information Bottleneck

**Mélanie Rey**
Department of Mathematics and Computer Science
University of Basel
`melanie.rey@unibas.ch`

**Volker Roth**
Department of Mathematics and Computer Science
University of Basel
`volker.roth@unibas.ch`

## Abstract

We present a reformulation of the information bottleneck (IB) problem in terms of copula, using the equivalence between mutual information and negative copula entropy. Focusing on the Gaussian copula we extend the analytical IB solution available for the multivariate Gaussian case to distributions with a Gaussian dependence structure but arbitrary marginal densities, also called meta-Gaussian distributions. This opens new possibles applications of IB to continuous data and provides a solution more robust to outliers.

## 1   Introduction

The information bottleneck method (IB) [1] considers the concept of relevant information in the data compression problem, and takes a new perspective to signal compression which was classically treated using rate distortion theory. The IB method formalizes the idea of relevance, or meaningful information, by introducing a relevance variable $Y$. The problem is then to obtain an optimal compression $T$ of the data $X$ which preserves a maximum of information about $Y$. Although the IB method beautifully formalizes the compression problem under relevance constraints, the practical solution of this problem remains difficult, particularly in high dimensions, since the mutual informations $I(X;T), I(Y;T)$ must be estimated. The IB optimization problem has no available analytical solution in the general case. It can be solved iteratively using the generalized Blahut-Arimoto algorithm which, however, requires us to estimate the joint distribution of the potentially high-dimensional variables $X$ and $Y$. A formal analysis of the difficulties of this estimation problem was conducted in [2]. In the continuous case, estimation of multivariate densities becomes arduous and can be a major impediment to the practical application of IB. A notable exception is the case of joint Gaussian $(X, Y)$ for which an analytical solution for the optimal representation $T$ exists [3]. The optimal $T$ is jointly Gaussian with $(X, Y)$ [4] and takes the form of a noisy linear projection to eigenvectors of the normalised conditional covariance matrix. The existence of an analytical solution opens new application possibilities and IB becomes practically feasible in higher dimensions [5]. Finding closed form solutions for other continuous distribution families remains an open challenge. The practical usefulness of the Gaussian IB (GIB), on the other hand, suffers from its missing flexibility and the statistical problem of finding a robust estimate of the joint covariance matrix of $(X, Y)$ in high-dimensional spaces.

Compression and relevance in IB are defined in terms of mutual information (MI) of two random vectors $V$ and $W$, which is defined as the reduction in the entropy of $V$ by the conditional entropy of $V$ given $W$. MI bears an interesting relationship to copulas: mutual information equals negative copula entropy [6]. This relation between two seemingly unrelated concepts might appear surpris-

ing, but it directly follows from the definition of a copula as the object that captures the "pure" dependency structure of random variables [7]: a multivariate distribution consists of univariate random variables related to each other by a dependence mechanism, and copulas provide a framework to separate the dependence structure from the marginal distributions. In this work we reformulate the IB problem for the continuous variables in terms of copulas and enlighten that IB is completely independent of the marginal distributions of $X, Y$. The IB problem in the continuous case is in fact to find the optimal copula (or dependence structure) of $T$ and $X$, knowing the copula of $X$ and the relevance variable $Y$. We focus on the case of Gaussian copula and on the consequences of the IB reformulation for the Gaussian IB. We show that the analytical solution available for GIB can naturally be extended to multivariate distributions with Gaussian copula and arbitrary marginal densities, also called *meta-Gaussian* densities. Moreover, we show that the GIB solution depends only a correlation matrix, and not on the variance. This allows us to use robust rank correlation estimators instead of unstable covariance estimators, and gives a robust version of GIB.

## 2 Information Bottleneck and Gaussian IB

### 2.1 General Information Bottleneck.

Consider two random variables $X$ and $Y$ with values in the measurable spaces $\mathcal{X}$ and $\mathcal{Y}$. Their joint distribution $p_{XY}(x, y)$ will also be denoted $p(x, y)$ for simplicity. We construct a compressed representation $T$ of $X$ that is most informative about $Y$ by solving the following variational problem:

$$\min_{p(t|x)} \mathcal{L} \mid \mathcal{L} \equiv I(X; T) - \beta I(T; Y), \tag{1}$$

where the Lagrange parameter $\beta > 0$ determines the trade-off between compression of $X$ and preservation of information about $Y$. Since the compressed representation is conditionally independent of $Y$ given $X$ as illustrated in Figure 1, to fully characterize $T$ we only need to specify its joint distribution with $X$, i.e. $p(x, t)$. No analytical solution is available for the general problem defined by (1) and this joint distribution must be calculated with an iterative procedure. In the case of discrete variables $X$ and $Y$, $p(x, t)$ is obtained iteratively by self-consistent determination of $p(t|x)$, $p(t)$ and $p(y|t)$ in the generalized Blahut-Arimoto algorithm. The resulting discrete $T$ then defines (soft) clusters of $X$. In the case of continuous $X$ and $Y$, the same set of self-consistent equations for $p(t|x)$, $p(t)$ and $p(y|t)$ are obtained. These equations also translate into two coupled eigenvector problems for $\partial \log p(x|t)/\partial t$ and $\partial \log p(y|t)/\partial t$, but a direct solution of these problems is very difficult in practice. However, when $X$ and $Y$ are jointly multivariate Gaussian distributed, this problem becomes analytically tractable.

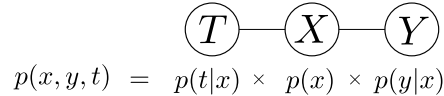

$$p(x, y, t) \quad = \quad p(t|x) \quad \times \quad p(x) \quad \times \quad p(y|x)$$

Figure 1: Graphical representation of the conditional independence structure of IB.

### 2.2 Gaussian IB.

Consider two joint Gaussian random vectors (rv) $X$ and $Y$ with zero mean:

$$(X, Y) \sim \mathcal{N}\left(0_{p+q}, \Sigma = \begin{pmatrix} \Sigma_x & \Sigma_{xy}^T \\ \Sigma_{xy} & \Sigma_y \end{pmatrix}\right), \tag{2}$$

where $p$ is the dimension of $X$, $q$ is the dimension of $Y$ and $0_{p+q}$ is the zero vector of dimension $p+q$. In [4] it is proved that the optimal compression $T$ is also jointly Gaussian with $X$ and $Y$. This implies that $T$ can be expressed as a noisy linear transformation of $X$:

$$T = AX + \xi, \tag{3}$$

where $\xi \sim \mathcal{N}(0_p, \Sigma_\xi)$ is independent of $X$ and $A \in \mathbb{R}^{p \times p}$. The minimization problem (1) is then reduced to solving:

$$\min_{A, \Sigma_\xi} \mathcal{L} | \mathcal{L} \equiv I(X;T) - \beta I(T;Y). \tag{4}$$

For a given trade-off parameter $\beta$, the optimal compression is given by $T \sim \mathcal{N}(0_p, \Sigma_t)$ with $\Sigma_t = A\Sigma_x A^T + \Sigma_\xi$ and the noise variance can be fixed to the identity matrix $\Sigma_\xi = I_p$, as shown in [3]. The transformation matrix $A$ is given by:

$$A = \begin{pmatrix} [0^T; \ldots; 0^T] & 0 \leq \beta \leq \beta_1^c \\ [\alpha_1 v_1^T; 0^T; \ldots, ; 0^T] & \beta_1^c \leq \beta \leq \beta_2^c \\ [\alpha_1 v_1^T; \alpha_2 v_2^T; 0^T; \ldots; 0^T] & \beta_2^c \leq \beta \leq \beta_3^c \\ \vdots & \end{pmatrix} \tag{5}$$

where $v_1^T, \ldots, v_p^T$ are left eigenvectors of $\Sigma_{x|y}\Sigma_x^{-1}$ sorted by their corresponding increasing eigenvalues $\lambda_1, \ldots, \lambda_p$. The critical $\beta$ values are $\beta_i^c = (1 - \lambda_i)^{-1}$, and the $\alpha_i$ coefficients are defined by $\alpha_i = \sqrt{\frac{\beta(1-\lambda_i)-1}{\lambda_i r_i}}$ with $r_i = v_i^T \Sigma_x v_i$. In the above, $0^T$ is a $p$-dimensional row vector and semicolons separate rows of $A$. We can see from equation (5) that the optimal projection of $X$ is a combination of weighted eigenvectors of $\Sigma_{x|y}\Sigma_x^{-1}$. The number of selected eigenvectors, and thus the effective dimension of $T$, depends on the parameter $\beta$.

## 3  Copula and Information Bottleneck

### 3.1  Copula and Gaussian copula.

A multivariate distribution consists of univariate random variables related to each other by a dependence mechanism. Copulas provide a framework to separate the dependence structure from the marginal distributions. Formally, a $d$-dimensional copula is a multivariate distribution function $C : [0,1]^d \to [0,1]$ with standard uniform margins. Sklar's theorem [7] states the relationship between copulas and multivariate distributions. Any joint distribution function $F$ can be represented using its marginal univariate distribution functions and a copula:

$$F(z_1, \ldots, z_d) = C(F_1(z_1), \ldots, F_d(z_d)). \tag{6}$$

If the margins are continuous, then this copula is unique. Conversely, if $C$ is a copula and $F_1, \ldots, F_d$ are univariate distribution functions, then $F$ defined as in (6) is a valid multivariate distribution function with margins $F_1, \ldots, F_d$. Assuming that $C$ has $d$-th order partial derivatives we can define the *copula density function*: $c(u_1, \ldots, u_d) = \frac{\partial C(u_1, \ldots, u_d)}{\partial u_1 \ldots \partial u_d}$, $u_1, \ldots, u_d \in [0,1]$, The density corresponding to (6) can then be rewritten as a product of the marginal densities and the copula density function: $f(z_1, \ldots, z_d) = c(F_1(z_1), \ldots, F_d(z_d)) \prod_{j=1}^d f_j(z_j)$.

Gaussian copulas constitute an important class of copulas. If $F$ is a Gaussian distribution $\mathcal{N}_d(\mu, \Sigma)$ then the corresponding $C$ fulfilling equation (6) is a Gaussian copula. Due to basic invariance properties (cf. [8]), the copula of $\mathcal{N}_d(\mu, \Sigma)$ is the same as the copula of $\mathcal{N}_d(0, P)$, where $P$ is the correlation matrix corresponding to the covariance matrix $\Sigma$. Thus a Gaussian copula is uniquely determined by a correlation matrix $P$ and we denote a Gaussian copula by $C_P$. Using equation (6) with $C_P$, we can construct multivariate distributions with arbitrary margins and a Gaussian dependence structure. These distributions are called *meta-Gaussian distributions*. Gaussian copulas conveniently have a copula density function:

$$c_P(u) = |P|^{-\frac{1}{2}} \exp\left\{ -\frac{1}{2} \Phi^{-1}(u)^T (P^{-1} - I) \Phi^{-1}(u) \right\}, \tag{7}$$

where $\Phi^{-1}(u)$ is a short notation for the univariate Gaussian quantile function applied to each component $\Phi^{-1}(u) = (\Phi^{-1}(u_1), \ldots, \Phi^{-1}(u_d))$.

### 3.2  Copula formulation of IB.

At the heart of the copula formulation of IB is the following identity: for a continuous random vector $Z = (Z_1, \ldots, Z_d)$ with density $f(z)$ and copula density $c_Z(u)$ the multivariate mutual information

or *multi-information* is the negative differential entropy of the copula density:

$$I(Z) \equiv D_{kl}(f(z) \parallel f_0(z)) = \int_{[0,1]^d} c_Z(u) \log c_Z(u) \mathrm{d}u = -H(c_Z), \tag{8}$$

where $u = (u_1, \ldots, u_d) \in [0,1]^d$, $D_{kl}$ denotes the Kullback-Leibler divergence, and $f_0(z) = f_1(z_1)f_2(z_2)\ldots f_d(z_d)$. For continuous multivariate $X$, $Y$ and $T$, equation (8) implies that:

$$I(X;T) = D_{kl}(f(x,t) \parallel f_0(x,t)) - D_{kl}(f(x)||f_0(x)) - D_{kl}(f(t)||f_0(t)),$$
$$= -H(c_{XT}) + H(c_X) + H(c_T),$$
$$I(Y;T) = -H(c_{YT}) + H(c_Y) + H(c_T),$$

where $c_{XT}$ is the copula density of the vector $(X_1, \ldots, X_p, T_1, \ldots, T_p)$. The above derivation then leads to the following proposition.

**Proposition 3.1.** Copula formulation of IB
*The Information Bottleneck minimization problem* (1) *can be reformulated as:*

$$\min_{c_{XT}} \mathcal{L} \mid \mathcal{L} = -H(c_{XT}) + H(c_X) + H(c_T) - \beta\{-H(c_{YT}) + H(c_Y) + H(c_T)\}. \tag{9}$$

The minimization problem defined in (1) is solved under the assumption that the joint distribution of $(X, Y)$ is known, this now translates in the assumption that the copula copula density $c_{XY}$ (and thus $c_X$) is assumed to be known. The density $c_T$ is entirely determined by $c_{XT}$, and using the conditional independence structure it is clear that $c_{YT}$ is also determined by $c_{XT}$ when $c_{XY}$ is known. Since the joint density of $(X, Y, T)$ decomposes as:

$$f(x,y,t) = f(t,y|x)f(x) = f(t|x)f(y|x)f(x), \tag{10}$$

the corresponding copula density then also decomposes as:

$$c_{XYT}(u_x, u_y, u_t) = R_{T|X}(u_x, u_t)R_{Y|X}(u_x, u_y)c_X(u_x), \tag{11}$$

where

$$R_{T|X}(u_x, u_t) = \frac{c_{XT}(u_x, u_t)}{c_X(u_x)}, \ u_x \in [0,1]^p, u_y \in [0,1]^q, u_t \in [0,1]^p, \tag{12}$$

as shown in [9]. We can finally rewrite the copula density of $(Y, T)$ as:

$$c_{YT}(u_y, u_t) = \int c_{XYT}(u_x, u_y, u_t)\mathrm{d}u_x = \int \frac{c_{XT}(u_x, u_t)c_{XY}(u_x, u_y)}{c_X(u_x)}\mathrm{d}u_x. \tag{13}$$

The IB optimization problem actually reduces to finding an optimal copula density $c_{XT}$. This implies that in order to construct the compression variable $T$, the only relevant aspect is the copula dependence structure between $X, T$ and $Y$.

## 4  Meta-Gaussian IB

### 4.1  Meta-Gaussian IB formulation.

The above reformulation of IB is of great practical interest when we focus on the special case of the Gaussian copula. The only known case for which a simple analytical solution to the IB problem exists is when $(X, Y)$ are joint Gaussians. Equation (9) shows that actually an optimal solution does not depend of the margins but only on the copula density $c_{XY}$. From this observation the idea naturally follows that an analytical solution should also exist for any joint distribution of $(X, Y)$ which has a Gaussian copula, and that regardless of its margins. We show below in proposition 4.1 that this is indeed the case. The notation $\tilde{X}$ and $\tilde{Y}$ is used to represent the normal scores:

$$\tilde{X} = (\Phi^{-1} \circ F_{X_1}(X_1), \ldots, \Phi^{-1} \circ F_{X_p}(X_p)). \tag{14}$$

Since copulas are invariant to strictly increasing transformations the normal scores have the same copulas as the original variables $X$ and $Y$.

**Proposition 4.1.** Optimality of meta-Gaussian IB
*Consider rv $X, Y$ with a Gaussian dependence structure and arbitrary margins:*

$$F_{X,Y}(x,y) \sim C_P(F_{X_1}(x_1), \ldots, F_{X_p}(x_p), F_{Y_1}(y_1), \ldots, F_{Y_q}(y_q)), \tag{15}$$

*where $F_{X_i}, F_{Y_i}$ are the marginal distributions of $X, Y$ and $C_P$ is a Gaussian copula parametrized by a correlation matrix $P$. Then the optimum of the minimization problem* (1) *is obtained for $T \in \mathcal{T}$, where $\mathcal{T}$ is the set of all rv $T$ such that $(X, Y, T)$ has a Gaussian copula and $T$ has Gaussian margins.*

Before proving proposition 4.1 we give a short lemma.

**Lemma 4.1.** $T \in \mathcal{T} \Leftrightarrow (\tilde{X}, \tilde{Y}, T)$ *are jointly Gaussian.*

*Proof.*    1. If $T \in \mathcal{T}$ then $(X, Y, T)$ has a Gaussian copula which implies that $(\tilde{X}, \tilde{Y}, T)$ also has a Gaussian copula. Since $\tilde{X}, \tilde{Y}, T$ all have normally distributed margins it follows that $(\tilde{X}, \tilde{Y}, T)$ has a joint Gaussian distribution.

2. If $(\tilde{X}, \tilde{Y}, T)$ are jointly Gaussian then $(\tilde{X}, \tilde{Y}, T)$ has a Gaussian copula which implies that $(X, Y, T)$ has again a Gaussian copula. Since $T$ has normally distributed margins, it follows that $T \in \mathcal{T}$.

$\square$

Proposition 4.1 can now be proven by contradiction.

*Proof of proposition 4.1.* Assume there exists $T^* \notin \mathcal{T}$ such that:

$$\mathcal{L}(X, Y, T^*) := I(X; T^*) - \beta I(Y; T^*) < \min_{p(t|x), T \in \mathcal{T}} I(X; T) - \beta I(T; Y) \tag{16}$$

Since $(\tilde{X}, \tilde{Y}, T)$ has the same copula as $(X, Y, T)$, we have that $I(\tilde{X}; T) = I(X; T)$ and $I(\tilde{Y}; T) = I(Y; T)$. Using Lemma 4.1 the right hand part of inequality (16) can be rewritten as :

$$\min_{p(t|x), T \in \mathcal{T}} \mathcal{L}(X, Y, T) = \min_{p(t|x), T \in \mathcal{T}} \mathcal{L}(\tilde{X}, \tilde{Y}, T) = \min_{p(t|\tilde{x}), (\tilde{X}, \tilde{Y}, T) \sim \mathcal{N}} \mathcal{L}(\tilde{X}, \tilde{Y}, T). \tag{17}$$

Combining equations (16) and (17) we obtain:

$$I(\tilde{X}; T^*) - \beta I(\tilde{Y}; T^*) < \min_{p(t|\tilde{x}), (\tilde{X}, \tilde{Y}, T) \sim \mathcal{N}} I(\tilde{X}; T) - \beta I(T; \tilde{Y}).$$

This is in contradiction with the optimality of Gaussian information bottleneck, which states that the optimal $T$ is jointly Gaussian with $(X, Y)$. Thus the optimum for meta-Gaussian $(X, Y)$ is attained for $T$ with normal margins such that $(X, Y, T)$ also is meta-Gaussian.

$\square$

**Corollary 4.1.** *The optimal projection $T^o$ obtained for $(\tilde{X}, \tilde{Y})$ is also optimal for $(X, Y)$.*

*Proof.* By the above we know that an optimal compression for $(X, Y)$ can be obtained in the set of variables $T$ such that $(\tilde{X}, \tilde{Y}, T)$ is jointly Gaussian, since $\tilde{\mathcal{L}} = \mathcal{L}$ it is clear that $T^o$ is also optimal for $(X, Y)$. $\square$

As a consequence of Proposition 4.1, for any random vector $(X, Y)$ having a Gaussian copula dependence structure, an optimal projection $T$ can be obtained by first calculating the vector of the normal scores $(\tilde{X}, \tilde{Y})$ and then computing $T = A\tilde{X} + \xi$. $A$ is here entirely determined by the covariance matrix of the vector $(\tilde{X}, \tilde{Y})$ which also equals its correlation matrix (the normal scores have unit variance by definition), and thus the correlation matrix $P$ parametrizing the Gaussian copula $C_P$. In practice the problem is reduced to the estimation the Gaussian copula of $(X, Y)$. In particular, for the traditional Gaussian case where $(X, Y) \sim \mathcal{N}(0, \Sigma)$, this means that we actually do not need to estimate the full covariance $\Sigma$ but only the correlations.

## 4.2 Meta-Gaussian mutual information.

The multi-information for a meta-Gaussian random vector $Z = (Z_1, \ldots, Z_d)$ with copula $C_{P_z}$ is:

$$I(Z) = I(\tilde{Z}) = -\tfrac{1}{2} \log |\text{cov}(\tilde{Z})| = -\tfrac{1}{2} \log |\Sigma_{\tilde{z}}| = -\tfrac{1}{2} \log |\text{corr}(\tilde{Z})| = -\tfrac{1}{2} \log |P_z|, \quad (18)$$

where $|.|$ denotes the determinant. A direct derivation of the multi-information for meta-Gaussian random variables is also given in the supplementary material. The mutual information between $X$ and $Y$ is then $I(X;Y) = -\tfrac{1}{2} \log |P| + \tfrac{1}{2} \log |P_x| + \tfrac{1}{2} \log |P_y|$, where $P = \begin{pmatrix} P_x & P_{yx} \\ P_{xy} & P_y \end{pmatrix}$. It is obvious that the formula for the meta-Gaussian is similar to the formula for the Gaussian case $I_{\text{Gauss}}(X;Y) = -\tfrac{1}{2} \log |\Sigma| + \tfrac{1}{2} \log |\Sigma_x| + \tfrac{1}{2} \log |\Sigma_y|$, but uses the correlation matrix parametrizing the copula instead of the data covariance matrix. The two formulas are equivalent when $X, Y$ are jointly Gaussian.

## 4.3 Semi-parametric copula estimation.

Semi-parametric copula estimation has been studied in [10], [11] and [12]. The main idea is to combine non-parametric estimation of the margins with a parametric copula model, in our case the Gaussian copulas family. If the margins $F_1, \ldots, F_d$ of a random vector $Z$ are known, $P$ can be estimated by the matrix $\hat{P}$ with elements given by:

$$\hat{P}_{(k,l)} = \frac{\tfrac{1}{n} \sum_{i=1}^{n} \Phi^{-1}(F_k(z_{ik})) \Phi^{-1}(F_l(z_{il}))}{\left[ \tfrac{1}{n} \sum_{i=1}^{n} \left[ \Phi^{-1}(F_k(z_{ik})) \right]^2 \tfrac{1}{n} \sum_{i=1}^{n} \left[ \Phi^{-1}(F_l(z_{il})) \right]^2 \right]^{1/2}}, \quad (19)$$

where $z_{ik}$ denotes the $i$-th observation of dimension $k$. $\hat{P}$ is assured to be positive semi-definite. If the margins are unknown we can instead use the rescaled empirical cumulative distributions:

$$\hat{F}_j(t) = \frac{n}{n+1} \left( \frac{1}{n} \sum_{i=1}^{n} \mathbb{I}_{z_{ij} \leq t} \right). \quad (20)$$

The estimator resulting from using the rescaled empirical distributions (20) in equation (19) is given in the following definition.

**Definition 4.1** (Normal scores rank correlation coefficient)**.** *The normal scores rank correlation coefficient is the matrix $\hat{P}^n$ with elements:*

$$\hat{P}^n_{(k,l)} = \frac{\sum_{i=1}^{n} \Phi^{-1}\left( \frac{R(z_{ik})}{n+1} \right) \Phi^{-1}\left( \frac{R(z_{il})}{n+1} \right)}{\sum_{i=1}^{n} \left( \Phi^{-1}\left( \frac{i}{n+1} \right) \right)^2}, \quad (21)$$

where $R(z_{ik})$ denotes the rank of the $i$-th observation for dimension $k$. Robustness properties of the estimator (21) have been studied in [13]. Using (21) we compute an estimate of the correlation matrix $P$ parametrizing $c_{XY}$ and obtain the transformation matrix $A$ as detailed in Algorithm 1.

---

**Algorithm 1** Construction of the transformation matrix $A$

---

1. Compute the normal scores rank correlation estimate $\hat{P}^n$ of the correlation matrix $P$ parametrizing $c_{XY}$:
**for** $k, l = 1, \ldots, p+q$ **do**

   Set the $(k,l)$-*th* element of $\hat{P}^n$ to $\frac{\sum_{i=1}^{n} \Phi^{-1}\left( \frac{R(z_{ik})}{n+1} \right) \Phi^{-1}\left( \frac{R(z_{il})}{n+1} \right)}{\sum_{i=1}^{n} \left( \Phi^{-1}\left( \frac{i}{n+1} \right) \right)^2}$ as in equation (21) and where

   the $i$-th row of $z$ is the concatenation of the $i$-th rows of $x$ and $y$: $z_{i*} = (x_{i*}, y_{i*}) \in \mathbb{R}^{p+q}$.
**end for**
2. Compute the estimated conditional covariance matrix of the normal scores: $\hat{\Sigma}_{\tilde{x}|\tilde{y}} = \hat{P}^n_x - \hat{P}^n_{xy} (\hat{P}^n_y)^{-1} \hat{P}^n_{yx}$.
3. Find the eigenvectors and eigenvalues of $\hat{\Sigma}_{\tilde{x}|\tilde{y}} (\hat{P}^n_x)^{-1}$.
4. Construct the transformation matrix $A$ as in equation (5).

---

# 5 Results

## 5.1 Simulations

We tested meta-Gaussian IB (MGIB) in two different setting, first when the data is Gaussian but contains outliers, second when the data has a Gaussian copula but non-Gaussian margins. We generated a training sample with $n = 1000$ observations of $X$ and $Y$ with dimensions fixed to $d_x = 15$ and $d_y = 15$. A covariance matrix was drawn from a Wishart distribution centered at a correlation matrix populated with a few high correlation values to ensure some dependency between $X$ and $Y$. This matrix was then scaled to obtain the correlation matrix parametrizing the copula. In the first setting the data was sampled with $\mathcal{N}(0, 1)$ margins. A fixed percentage of outliers, $8\%$, was then introduced to the sample by randomly drawing a row and a column in the data matrix and replacing the current value with a random draw from the set $[-6, -3] \cup [3, 6]$. In the second setting data points were drawn from meta-Gaussian distributions with three different type of margins: Student with $df = 4$, exponential with $\lambda = 1$, and beta with $\alpha_1 = 0.5 = \alpha_2$. For each training sample two projection matrices $A_G$ and $A_C$ were computed, $A_G$ was calculated based on the sample covariance $\hat{\Sigma}^n$ and $A_C$ was obtained using the normal scores rank correlation $\hat{P}^n$. The compression quality of the projection was then tested on a test sample of $n = 10'000$ observations generated independently from the same distribution (without outliers). Each experiment was repeated 50 times. Figure 2 shows the information curves obtained by varying $\beta$ from 0.1 to 200. The mutual informations $I(X; T)$ and $(Y; T)$ can be reliably estimated on the test sample using (18) and (21). The information curves start with a very steep slope, meaning that a small increase in $I(X; T)$ leads to a significant increase in $I(Y; T)$, and then slowly saturate to reach their asymptotic limit in $I(Y; T)$. The best information curves are situated in the upper left corner of the figure, since for a fixed compression value $I(X; T)$ we want to achieve the highest relevant information content $(I; T)$. We clearly see in Figure 2 that MGIB consistently outperforms GIB in that it achieves higher compression rates.

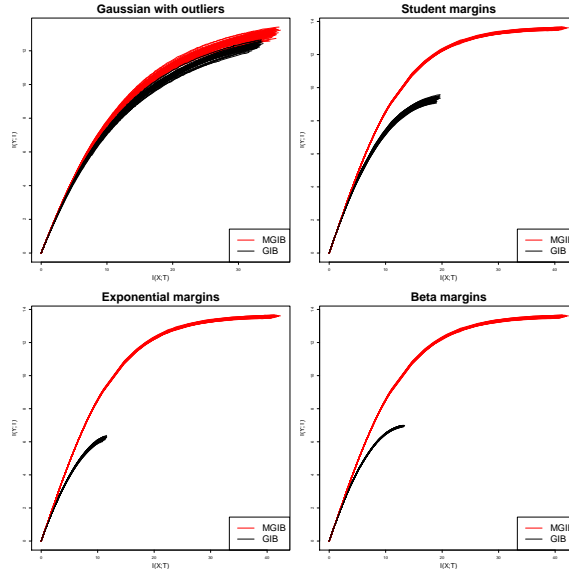

Figure 2: Information curves for Gaussian data with outliers, data with Student, Exponential and Beta margins. Each panel shows 50 curves obtained for repetitions of the MGIB (red) and the GIB (black). The curves stop when they come close to saturation. For higher values of $\beta$ the information $I(X; T)$ would continue to grow while $I(Y; T)$ would reach its limit leading to horizontal lines, but such high beta values lead to numerical instability. Since GIB suffers from a model mismatch problem when the margins are not Gaussian, the curves saturate for smaller values of $I(Y; T)$.

## 5.2 Real data

We further applied MGIB to the *Communities and Crime* data set from the UCI repository [1]. The data set contains observations of predictive and target variables. After removing missing values we retained $n = 2195$ observations. In a pre-processing step we selected the $d_x = 10$ dimensions with the strongest absolute rank correlation to one of the relevance variables. Plotting empirical information curves as in the synthetic examples above was impossible, because even for this setting with drastically decreased dimensionality all mutual information estimates we tried (including the nearest-neighbor graph method in [14]) were too unstable to draw empirical information curves. To still give a graphical representation of our results we show in Figure 3 non-parametric density estimates of the one dimensional compression $T$ split in 5 groups according to corresponding values of the first relevance variable. We used GIB, MGIB and Principal Component analysis (PCA) to reduce $X$ to a 1-dimensional variable. For PCA this is the first principal component, for GIB and MGIB we independently selected the highest value of $\beta$ leading to a 1-dimensional compression. It is obvious from Figure 3 that the one-dimensional MGIB compression nicely separates the different target classes, whereas the GIB and PCA projections seem to contain much less information about the target variable. We conclude that similar to our synthetic examples above, the MGIB compression contains more information about the relevance variable than GIB at the same compression rate.

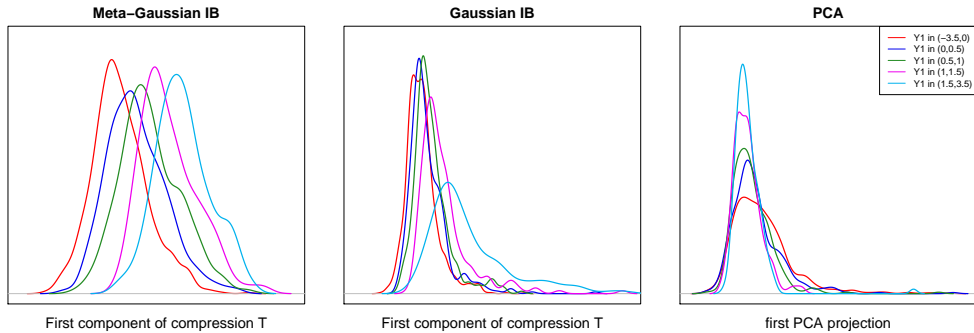

Figure 3: Parzen density estimates of the univariate projection of $X$ split in 5 groups according to values of the first relevance variable. We see more separation between groups for MGIB than for GIB or PCA, which indicates that the projection is more informative about the relevance variable.

## 6 Conclusion

We present a reformulation of the IB problem in terms of copula which gives new insights into data compression with relevance constraints and opens new possible applications of IB for continuous multivariate data. Meta-Gaussian IB naturally extends the analytical solution of Gaussian IB to multivariate distributions with Gaussian copula and arbitrary marginal density. It can be applied to any type of continuous data, provided the assumption of a Gaussian dependence structure is reasonable, in which case the optimal compression can easily be obtained by semi-parametric copula estimation. Simulated experiments showed that MGIB clearly outperforms GIB when the marginal densities are not Gaussian, and even in the Gaussian case with a tiny amount of outliers MGIB has been shown to significantly benefit from the robustness properties of rank estimators. In future work, it would be interesting to see if the copula formulation of IB admits analytical solutions for other copula families.

### Acknowledgments

M. Rey is partially supported by the Swiss National Science Foundation, grant CR32I2 127017 / 1.

[1] http://archive.ics.uci.edu/ml/

# References

[1] N. Tishby, F.C. Pereira, and W. Bialek. The information bottleneck method. *The 37th annual Allerton Conference on Communication, Control, and Computing*, (29-30):368–377, 1999.

[2] O. Shamir, S. Sabato, and N. Tishby. Learning and generalization with the information bottleneck. *Theor. Comput. Sci.*, 411(29-30):2696–2711, 2010.

[3] G. Chechik, A. Globerson, N. Tishby, and Y. Weiss. Information bottleneck for Gaussian variables. *Journal of Machine Learning Research*, 6:165–188, 2005.

[4] A. Globerson and N. Tishby. On the optimality of the Gaussian information bottleneck curve. *Hebrew University Technical Report*, 2004.

[5] R.M. Hecht, E. Noor, and N. Tishby. Speaker recognition by Gaussian information bottleneck. *INTER-SPEECH*, pages 1567–1570, 2009.

[6] J. Ma and Z. Sun. Mutual information is copula entropy. *arXiv:0808.0845v1*, 2008.

[7] A. Sklar. Fonctions de répartition à n dimensions et leurs marges. *Publications de l'Institut de Statistique de l'Université de Paris*, 8:229–231, 1959.

[8] A. J. McNeil, R. Frey, and P. Embrechts. *Quantitative Risk Management*. Princeton Series in Finance. Princeton University Press, 2005.

[9] G. Elidan. Copula bayesian networks. *Proceedings of the Neural Information Processing Systems (NIPS)*, 2010.

[10] C. Genest, K. Ghoudhi, and L.P. Rivet. A semiparametric estimation procedure of dependence parameters in multivariate families of distributions. *Biometrika*, 82(3):543–552, 1995.

[11] H. Tsukahara. Semiparametric estimation in copula models. *The Canadian Journal of Statistics*, 33(3):357–375, 2005.

[12] Peter D. Hoff. Extending the rank likelihood for semiparametric copula estimation. *Annals of Applied Statistics*, 1(1):273, 2007.

[13] K. Boudt, J. Cornelissen, and C. Croux. The gaussian rank correlation estimator: Robustness properties. *Statistics and Computing*, 22:471–483, 2012.

[14] D. Pál, B. Póczos, and C. Szepesvári. Estimation of Rényi entropy and mutual information based on generalized nearest-neighbor graphs. *Proceedings of the Neural Information Processing Systems (NIPS)*, 2010.

